# Factor Modeling for Advertisement Targeting

**Ye Chen**[*]
eBay Inc.
yechen1@ebay.com

**Michael Kapralov**
Stanford University
kapralov@stanford.edu

**Dmitry Pavlov**[†]
Yandex Labs
dmitry-pavlov@yandex-team.ru

**John F. Canny**
University of California, Berkeley
jfc@cs.berkeley.edu

## Abstract

We adapt a probabilistic latent variable model, namely GaP (Gamma-Poisson) [6], to ad targeting in the contexts of sponsored search (SS) and behaviorally targeted (BT) display advertising. We also approach the important problem of ad positional bias by formulating a one-latent-dimension GaP factorization. Learning from click-through data is intrinsically large scale, even more so for ads. We scale up the algorithm to terabytes of real-world SS and BT data that contains hundreds of millions of users and hundreds of thousands of features, by leveraging the scalability characteristics of the algorithm and the inherent structure of the problem including data sparsity and locality. Specifically, we demonstrate two somewhat orthogonal philosophies of scaling algorithms to large-scale problems, through the SS and BT implementations, respectively. Finally, we report the experimental results using Yahoo's vast datasets, and show that our approach substantially outperform the state-of-the-art methods in prediction accuracy. For BT in particular, the ROC area achieved by GaP is exceeding 0.95, while one prior approach using Poisson regression [11] yielded 0.83. For computational performance, we compare a single-node sparse implementation with a parallel implementation using Hadoop MapReduce, the results are counterintuitive yet quite interesting. We therefore provide insights into the underlying principles of large-scale learning.

## 1 Introduction

Online advertising has become the cornerstone of many sustainable business models in today's Internet, including search engines (e.g., Google), content providers (e.g., Yahoo!), and social networks (e.g., Facebook). One essential competitive advantage, over traditional channels, of online advertising is that it allows for targeting. The objective of ad targeting is to select most relevant ads to present to a user based on contextual and prior knowledge about this user. The relevance measure or response variable is typically click-through rate (CTR), while explanatory variables vary in different application domains. For instance, sponsored search (SS) [17] uses query, content match [5] relies on page content, and behavioral targeting (BT) [11] leverages historical user behavior. Nevertheless, the training data can be generally formed as a user-feature matrix of event counts, where the feature dimension contains various events such as queries, ad clicks and views. This characterization of data naturally leads to our adoption of the family of latent variable models [20, 19, 16, 18, 4, 6], which have been quite successfully applied to text and image corpora. In general, the goal of latent variable models is to discover statistical structures (factors) latent in the data, often with dimensionality reduction, and thus to generalize well to unseen examples. In particular, our choice of Gamma-Poisson (GaP) is theoretically as well as empirically motivated, as we elaborate in Section 2.2.

---

[*][†]This work was conducted when the authors were at Yahoo! Labs, 701 First Ave, Sunnyvale, CA 94089.

Sponsored search involves placing textual ads related to the user query alongside the algorithmic search results. To estimate ad relevance, previous approaches include similarity search [5], logistic regression [25, 8], classification and online learning with perceptron [13], while primarily in the original term space. We consider the problem of estimating CTR of the form $p(\text{click}|\text{ad}, \text{user}, \text{query})$, through a factorization of the user-feature matrix into a latent factor space, as derived in Section 2.1. SS adopts the keyword-based pay-per-click (PPC) advertising model [23]; hence the accuracy of CTR prediction is essential in determining the ad's ranking, placement, pricing, and filtering [21].

Behavioral targeting leverages historical user behavior to select relevant ads to display. Since BT does not primarily rely on contextual information such as query and page content; it makes an enabling technology for display (banner) advertising where such contextual data is typically unavailable, such as reading an email, watching a movie, instant messaging, and at least from the ad's side. We consider the problem of predicting CTR of the form $p(\text{click}|\text{ad}, \text{user})$. The question addressed by the state-of-the-art BT is instead that of predicting the CTR of an ad in a given category (e.g., Finance and Technology) or $p(\text{click}|\text{ad-category}, \text{user})$, by fitting a sign-constrained linear regression with categorized features [12] or a non-negative Poisson regression with granular features [11,10,7]. Ad categorization is done by human labeling and thus expensive and error-prone. One of the major advantages of GaP is the ability to perform granular or per-ad prediction, which is infeasible by the previous BT technologies due to scalability issues (e.g., a regression model for each category).

## 2  GaP model

GaP is a generative probabilistic model, as graphically represented in Figure 1. Let $F$ be an $n \times m$ data matrix whose element $f_{ij}$ is the observed count of event (or feature) $i$ by user $j$. $Y$ is a matrix of expected counts with the same dimensions as $F$. $F$, element-wise, is naturally assumed to follow Poisson distributions with mean parameters in $Y$ respectively, i.e., $F \sim \text{Poisson}(Y)$. Let $X$ be a $d \times m$ matrix where the column vector $\mathbf{x}_j$ is a low-dimensional representation of user $j$ in a latent space of "topics". The element $x_{kj}$ encodes the "affinity" of user $j$ to topic $k$ as the total number of occurrences of all events contributing to topic $k$. $\Lambda$ is an $n \times d$ matrix where the column $\Lambda_k$ represents the $k$th topic as a vector of event probabilities $p(i|k)$, that is, a multinomial distribution of event counts conditioned on topic $k$. Therefore, the Poisson mean matrix $Y$ has a linear parameterization with $\Lambda$ and $X$, i.e., $Y = \Lambda X$. GaP essentially yields an approximate factorization of the data matrix into two matrices with a low inner dimension $F \approx \Lambda X$. The approximation has an appealing interpretation column-wise $\mathbf{f} \approx \Lambda \mathbf{x}$, that is, each user vector $\mathbf{f}$ in event space is approximated by a linear combination of the column vectors of $\Lambda$, weighted by the topical mixture $\mathbf{x}$ for that user. Since by design $d \ll n, m$, the model matrix $\Lambda$ shall capture significant statistical (topical) structure hidden in the data. Finally, $x_{kj}$ is given a gamma distribution as an empirical prior. The generative process of an observed event-user count $f_{ij}$ follows:

1. Generate $x_{kj} \sim \text{Gamma}(\alpha_k, \beta_k), \forall k$.
2. Generate $y_{ij}$ occurrences of event $i$ from a mixture of $k$ $\text{Multinomial}(p(i|k))$ with outcome $i$, i.e., $y_{ij} = \Lambda_i \mathbf{x}_j$ where $\Lambda_i$ is the $i$th row vector of $\Lambda$.
3. Generate $f_{ij} \sim \text{Poisson}(y_{ij})$.

The starting point of the generative process is a gamma distribution of $x$, with pdf

$$p(x) = \frac{x^{\alpha-1} \exp(-x/\beta)}{\beta^\alpha \Gamma(\alpha)} \text{ for } x > 0 \text{ and } \alpha, \beta > 0. \tag{1}$$

It has a shape parameter $\alpha$ and a scale parameter $\beta$. Next, from the latent random vector characterizing a user $\mathbf{x}$, we derive the expected count vector $\mathbf{y}$ for the user as follows:

$$\mathbf{y} = \Lambda \mathbf{x}. \tag{2}$$

The last stochastic process is a Poisson distribution of the observed count $f$ with the mean value $y$,

$$p(f) = \frac{y^f \exp(-y)}{f!} \text{ for } f \geq 0. \tag{3}$$

The data likelihood for a user generated as described above is

$$\prod_{i=1}^{n} \frac{y_i^{f_i} \exp(-y_i)}{f_i!} \prod_{k=1}^{d} \frac{(x_k/\beta_k)^{\alpha_k-1} \exp(-x_k/\beta_k)}{\beta_k \Gamma(\alpha_k)}, \tag{4}$$

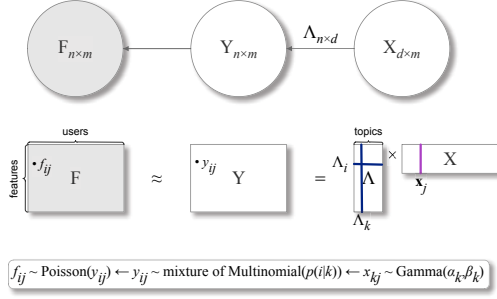 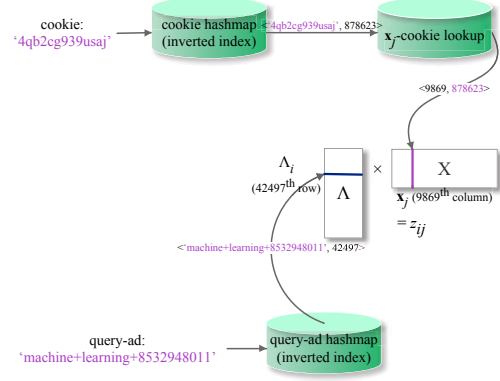

Figure 1: GaP graphical model          Figure 2: GaP online prediction

where $y_i = \Lambda_i \mathbf{x}$. And the log likelihood reads

$$\ell = \sum_i \left(f_i \log y_i - y_i - \log f_i!\right) + \sum_k \left[(\alpha_k - 1)\log x_k - x_k \beta_k + \alpha_k \log(\beta_k) - \log \Gamma(\alpha_k)\right] \quad (5)$$

Given a corpus of user data $F = (\mathbf{f}_1 \cdots \mathbf{f}_j \cdots \mathbf{f}_m)$, we wish to find the maximum likelihood estimates (MLE) of the model parameters $(\Lambda, X)$. Based on an elegant multiplicative recurrence developed by Lee and Seung [22] for NMF, the following EM algorithm was derived in [6]:

$$\text{E-step:} \quad x_{kj} \leftarrow x_{kj} \frac{\sum_i \left(f_{ij} \lambda_{ik} / y_{ij}\right) + (\alpha_k - 1)/x_{kj}}{\sum_i \lambda_{ik} + 1/\beta_k} \quad (6)$$

$$\text{M-step:} \quad \lambda_{ik} \leftarrow \lambda_{ik} \frac{\sum_j f_{ij}\overline{x}_{kj}/\overline{y}_{ij}}{\sum_j \overline{x}_{kj}} \quad (7)$$

## 2.1 Two variants for CTR prediction

The standard GaP model fits discrete count data. We now describe two variant derivations for predicting CTR. The first approach is to predict clicks and views independently, and then to construct the unbiased estimator of CTR, typically with Laplacian smoothing:

$$\text{CTR}_{\text{ad}(i)j} = \frac{\Lambda_{\text{click}(i)}\mathbf{x}_j + \alpha}{\Lambda_{\text{view}(i)}\mathbf{x}_j + \beta} \quad (8)$$

where $\text{click}(i)$ and $\text{view}(i)$ are the indices corresponding to the click/view pair of ad feature $i$, respectively, by user $j$; $\alpha$ and $\beta$ are smoothing constants.

The second idea is to consider the relative frequency of counts, particularly the number of clicks relative to the number of views for the events of interest. Formally, let $F$ be a matrix of observed click counts and $Y$ be a matrix of the corresponding expected click counts. We further introduce a matrix of observed views $V$ and a matrix of click probabilities $Z$, and define the link function:

$$F \gg Y = V \cdot Z = V \cdot (\Lambda X) \quad (9)$$

where '·' denotes element-wise matrix multiplication. The linear predictor $Z = \Lambda X$ now estimates CTR directly, and is scaled by the observed view counts $V$ to obtain the expected number of clicks $Y$. The Poisson assumption is only given to the click events $F$ with the mean parameters $Y$. Given a number of views $v$ and the probability of click for a single view or CTR, a more natural stochastic model for click counts is $\text{Binomial}(v, \text{CTR})$. But since in ad's data the number of views is sufficiently large and CTR is typically very small, the binomial converges to $\text{Poisson}(v \cdot \text{CTR})$.

Given the same form of log likelihood in Eq. (5) but with the extended link function in Eq. (9), we derive the following EM recurrence:

$$\text{E-step:} \quad x_{kj} \leftarrow x_{kj} \frac{\sum_i \left(f_{ij}\lambda_{ik}/z_{ij}\right) + (\alpha_k - 1)/x_{kj}}{\sum_i (v_{ij}\lambda_{ik}) + 1/\beta_k} \quad (10)$$

$$\text{M-step:} \quad \lambda_{ik} \leftarrow \lambda_{ik} \frac{\sum_j \left(f_{ij}\overline{x}_{kj}/\overline{z}_{ij}\right)}{\sum_j (v_{ij}\overline{x}_{kj})} \quad (11)$$

## 2.2 Rationale for GaP model

GaP is a generative probabilistic model for discrete data (such as texts). Similar to LDA (latent Dirichlet allocation) [4], GaP represents each sample (document or in this case a user) as a mixture of topics or interests. The latent factors in these models are non-negative, which has proved to have several practical advantages. First of all, texts arguably do comprise passages of prose on specific topics, whereas negative factors have no clear interpretation. Similarly, users have occasional interests in particular products or groups of products and their click-through propensity will dramatically increase for those products. On the other hand "temporary avoidance" of a product line is less plausible, and one clearly cannot have negative click-through counts which would be a consequence of allowing negative factors. A more practical aspect of non-negative factor models is that weak factor coefficients are driven to zero, especially when the input data is itself sparse; and hence the non-zeros will be much more stable, and cross-validation error much lower. This helps to avoid overfitting, and a typical LDA or GaP model can be run with high latent dimensions without overfitting, e.g., with 100 data measurements per user; one factor of a 100-dimensional PCA model will essentially be a (reversible) linear transformation of the input data. On the choice of GaP vs. LDA, the models are very similar, however there is a key difference. In LDA, the choice of latent factor is made independently word-by-word, or in the BT case, ad view by ad view. In GaP however, it is assumed that several items are chosen from each latent factor, i.e., that interests are locally related. Hence GaP uses gamma priors which include both shape and scale factors. The scale factors provide an estimated count of the number of items drawn from each latent factor. Another reason for our preference for GaP in this application is its simplicity. While LDA requires application of transcendental functions across the models with each iteration (e.g., $\Psi$ function in Equation (8) of [4]), GaP requires only basic arithmetic. Apart from transcendentals, the numbers of arithmetic operations of the two methods on same-sized data are identical. While we did not have the resources to implement LDA at this scale in addition to GaP, small-scale experiments showed identical accuracy. So we chose GaP for its speed and simplicity.

# 3 Sponsored search

We apply the second variant of GaP or the CTR-based formulation to SS CTR prediction, where the factorization will directly yield a linear predictor of CTR or $p(\text{click}|\text{ad}, \text{user}, \text{query})$, as in Eq. (9). Based on the structure of the SS click-through data, specifically the dimensionality and the user data locality, the deployment of GaP for SS involves three processes: (1) offline training, (2) offline user profile updating, and (3) online CTR prediction, as elaborated below.

## 3.1 The GaP deployment for SS

**Offline training.** First, given the observed click counts $F$ and view counts $V$ obtained from a corpus of historical user data, we derive $\Lambda$ and $X$ using the CTR-based GaP algorithm in Eqs. (10) and (11). Counts are aggregated over a certain period of time (e.g., one month) and for a feature space to be considered in the model. In SS, the primary feature type is the query-ad pair (noted as QL for query-linead, where linead refers to a textual ad) since it is the response variable of which the CTR is predicted. Other features can also be added based on their predicting capabilities, such as query term, linead term, ad group, and match type. This will effectively change the per-topic feature mixture in $\Lambda$ and possibly the per-user topic mixture in $X$, with the objective of improving CTR prediction by adding more contextual information. In prediction though, one only focuses on the blocks of QL features in $\Lambda$ and $Z$. In order for the model matrix $\Lambda$ to capture the corpus-wide topical structure, the entire user corpus should be used as training set.

**Offline user profile updateing.** Second, given the derived model matrix $\Lambda$, we update the user profiles $X$ in a distributed and data-local fashion. This updating step is necessary for two reasons. (1) User space is more volatile relative to feature space, due to cookie churn (fast turnover) and user's interests change over time. To ensure the model to capture the latest user behavioral pattern and to have high coverage of users, one needs to refresh the model often, e.g., on a daily basis. (2) Retraining the model from scratch is relatively expensive, and thus impractical for frequent model refresh. However, partial model refresh, i.e., updating $X$, has a very efficient and scalable solution which works as follows. Once a model is trained on a full corpus of user data, it suffices to keep only $\Lambda$, the model matrix so named. $\Lambda$ contains the global information of latent topics in the form

of feature mixtures. We then distribute $\Lambda$ across servers with each randomly bucketized for a subset of users. Note that this bucketization is exactly how production ad serving works. With the global $\Lambda$ and the user-local data $F$ and $V$, $X$ can be computed using E-step recurrence only. According to Eq. (10), the update rule for a given user $\mathbf{x}_j$ only involves the data for that user and the global $\Lambda$. Moreover, since $\Lambda$ and a local $X$ usually fit in memory, we can perform successive E-steps to converge $X$ within an order of magnitude less amount of time comparing with a global E-step. Notice that the multiplicative factor in E-step depends on $x_{kj}$, the parameter being updated, thus consecutive E-steps will indeed advance convergence.

**Online CTR prediction.** Finally, given the global $\Lambda$ and a local $X$ learned and stored in each server, the expected CTR for a user given a QL pair or $p(\text{click}|\text{QL}, \text{user})$ is computed online as follows. Suppose a user issues a query, a candidate set of lineads is retrieved by applying various matching algorithms. Taking the product of these lineads with the query gives a set of QLs to be scored. One then extracts the row vectors from $\Lambda$ corresponding to the candidate QL set to form a smaller block $\Lambda^{\text{mat}}$, and looks up the column vector $\mathbf{x}_j$ for that user from $X$. The predicted CTRs are obtained by a matrix-vector multiplication $\mathbf{z}_j^{\text{mat}} = \Lambda^{\text{mat}}\mathbf{x}_j$. The online prediction deployment is schematically shown in Figure 2.

## 3.2 Positional normalization

Our analysis so far has been abstracted from another essential factor, that is, the position of an ad impression on a search result page. It is known intuitively and empirically that ad position has a significant effect on CTR [24, 14]. In this section we treat the positional effect in a statistically sound manner.

The observed CTR actually represents a conditional probability $p(\text{click}|\text{position})$. We wish to learn a CTR normalized by position, i.e., "scaled" to a same presentation position, in order to capture the probability of click regardless of where the impression is shown. To achieve positional normalization, we assume the following Markov chain: (1) viewing an ad given its position, and then (2) clicking the ad given a user actually views the ad; thus

$$p(\text{click}|\text{position}) = p(\text{click}|\text{view})p(\text{view}|\text{position}), \tag{12}$$

where "view" is the event of a user voluntarily examining an ad, instead of an ad impression itself. Eq. (12) suggests a factorization of a matrix of observed CTRs into two vectors. As it turns out, to estimate the positional prior $p(\text{view}|\text{position})$ we can apply a special GaP factorization with one inner dimension. The data matrices $F$ and $V$ are now feature-by-position matrices, and the inner dimension can be interpreted as the topic of physically viewing.

In both training and evaluation, one shall use the position-normalized CTR, i.e., $p(\text{click}|\text{view})$. First, the GaP algorithm for estimating positional priors is run on the observed click and view counts of (feature, position) pairs. This yields a row vector of positional priors $\mathbf{x}^{\text{pos}}$. In model training, each ad view occurrence is then normalized (multiplied) by the prior $p(\text{view}|\text{position})$ for the position where the ad is presented. For example, the *a priori* CTR of a noticeable position (e.g., ov-top+1 in Yahoo's terminology meaning the North 1 position in sponsored results) is typically higher than that of an obscure position (e.g., ov-bottom+2) by a factor of up to 10. An observed count of views placed in ov-top+1 thus has a greater normalized count than that in ov-bottom+2. This normalization effectively asserts that, given a same observed (unnormalized) CTR, an ad shown in an inferior position has a higher click probability *per se* than the one placed in a more obvious position. The same view count normalization should also be applied during offline evaluation. In online prediction, however, we need CTR estimates unbiased by positional effect in order for the matching ads to be ranked based on their qualities (clickabilities). The linear predictor $Z = \Lambda X$ learned from a position-normalized training dataset gives exactly the position-unbiased CTR estimation. In other words, we are hypothesizing that all candidate ads are to be presented in a same imaginary position. For an intuitive interpretation, if we scale positional priors so that the top position has a prior of 1, i.e., $x^{\text{pos}}_{\text{ov-top+1}} = 1$, all ads are normalized to that top position.

Another view of the positional prior model we use is an examination model [25], that is, the probability of clicking on an ad is the product of a positional probability and a relevance-based probability which is independent of position. This model is simple and easy to solve for using maximum likelihood as explained above. This model is not dependent on the content of ads higher up on the search page, as for example the cascade [14] or DBN models [9]. These models are appropriate

for search results where users have a high probability of clicking on one of the links. However, for ads, the probability of clicking on ad links is extremely low, usually a fraction of a percent. Thus the effects of higher ads is a product of factors which are extremely close to one. In this case, the DBN positional prior reduces to a negative exponential function which is a good fit to the empirical distribution found from the examination model.

### 3.3 Large-scale implementation

**Data locality.** Recall that updating $X$ after a global training is distributed and only involves E-steps using user-local data. In fact, this data locality can also be leveraged in training. More precisely, Eq. (10) suggests that updating a user profile vector $\mathbf{x}_j$ via E-step only requires that user's data $\mathbf{f}_j$ and $\mathbf{v}_j$ as well as the model matrix $\Lambda$. This computation has a very small memory footprint and typically fits in L1 cache. On the other hand, updating each single value in $\Lambda$ as in Eq. (11) for M-step requires a full pass over the corpus (all users' data) and hence more expensive. To better exploit the data locality present in E-step, we alternate 3 to 10 successive E-steps with one M-step.

We also observe that M-step involves summations over $j \leq m$ users, for both the numerator and the denominator in Eq. (11). Both summing terms ($f_{ij}\overline{x}_{kj}/\overline{z}_{ij}$ and $v_{ij}\overline{x}_{kj}$) only requires data that is available locally (in memory) right after the E-step for user $j$. Thus the summations for M-step can be computed incrementally along with the E-step recurrence for each user. As thus arranged, an iteration of 3-10 E-steps combined with one M-step only requires a single pass over the user corpus.

**Data sparsity.** The multiplicative recurrence exploits data sparsity very well. Note that the inner loops of both E-step and M-step involve calculating the ratio $f_{ij}/z_{ij}$. Since $f$ is a count of very rare click events, one only needs to compute $z$ when the corresponding $f$ is non-zero. Let $N_c$ be the total number of non-zero $f$ terms or distinct click events over all users. For each non-zero $f_{ij}$, computing $z_{ij} = \Lambda_i \mathbf{x}_j$ dot-product takes $d$ multiplications. Thus the numerators of both E-step and M-step have a complexity of $O(N_c d)$. Both denominators have a complexity of $O(N_v)$, where $N_v$ is the total number of non-zero $v$ terms. The final divisions to compute the multiplicative factors in one outer loop over topics take $O(d)$ time (the other outer loop over $m$ or $n$ has already been accounted for by both $N_c$ and $N_v$). Typically, we have $N_v \gg N_c \gg m > n \gg d$. Thus the smoothed complexity [26] of offline training is $O(N_v d r)$, where $r$ is the number of EM iterations and $r = 20$ suffices for convergence.

**Scalability.** Now that we have reached an algorithm of linear complexity $O(N_v d r)$ with various implementation tricks as just described. We now illustrate the scalability of our algorithm by the following run-time analysis. The constant factor of the complexity is 4, the number of division terms in the recurrence formulae. Suppose the entire Yahoo's user base of SS contains about 200 million users. A $1/16$ sample (32 out of 512 buckets) gives around 10 million user. Further assume 100 distinct ad views on average per user and an inner dimension of 10, thus the total number of operations is $4 \times 10^{10}$ for one iteration. The model converges after 15-20 iterations. Our single-machine implementation with sparse matrix operations (which are readily available in MATLAB [2] and LAPACK [3]) gives above 100 Mflops, hence it takes 1.6-2.2 hours to train a model.

So far, we have demonstrated one paradigm of scaling up, which focuses on optimizing arithmetic operations, such as using sparse matrix multiplication in the innermost loop. Another paradigm is through large-scale parallelization, such as using a Hadoop [1] cluster, as we illustrate in the BT implementation in Section 4.1.

### 3.4 Experiments

We have experimented with different feature types, and found empirically the best combination is query-linead (QL), query term (QT), and linead term (LT). A QL feature is a product of query and linead. For QTs, queries are tokenized with stemming and stopwords removed. For LTs, we first concatenate the title, short description, and description of a linead text, and then extract up to 8 foremost terms. The dataset was obtained from 32 buckets of users and covering a one-month period, where the first three weeks forms the training set and the last week was held out for testing. For feature selection, we set the minimum frequency to 30 to be included for all three feature types, which yielded slightly above 1M features comprised of 700K QLs, 175K QTs, and 135K LTs. We also filtered out users with a total event count below 10, which gave 1.6M users. We used a latent

dimension of 10, which was empirically among the best while computationally favorable. For the gamma prior on $X$, we fixed the shape parameter $\alpha$ to $1.45$ and the scale parameter $\beta$ to $0.2$ across all latent topics for model training; and used a near-zero prior for positional prior estimation.

We benchmarked our GaP model with two simple baseline predictors: (1) Panama score (historical COEC defined as the ratio of the observed clicks to the expected clicks [9]), and (2) historical QL CTR normalized by position. The experimental results are plotted in Figure 3, and numerically summarized in Tables 1 and 2. A click-view ROC curve plots the click recall vs. the view recall, from the testing examples ranked in descending order of predicted CTR. A CTR lift curve plots the relative CTR lift vs. the view recall. As the results show, historical QL CTR is a fair predictor relative to Panama score. The GaP model yielded a ROC area of $0.82$ or $2\%$ improvement over historical QL CTR, and a $68\%$ average CTR lift over Panama score at the 5-20% view recall range.

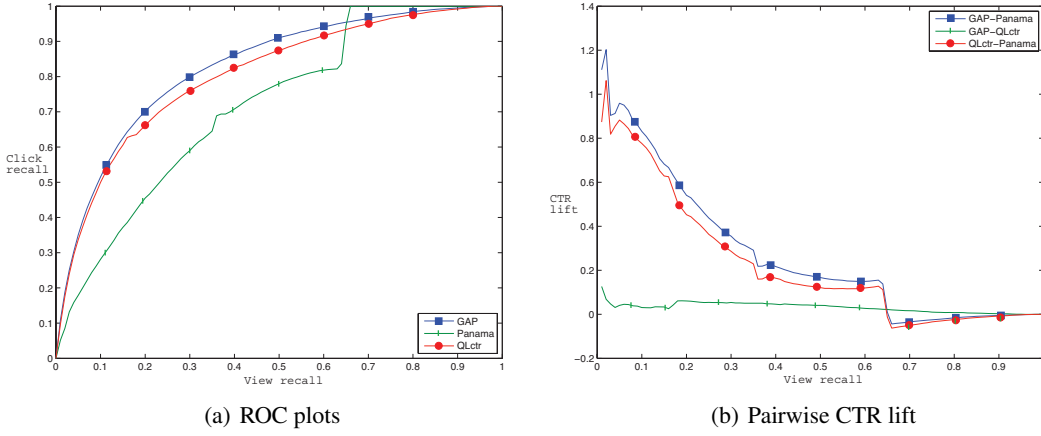

(a) ROC plots          (b) Pairwise CTR lift

Figure 3: Model performance comparison among (1) GaP using QL-QT-LT, (2) Panama score predictor, and (3) historical QL-CTR predictor.

Table 1: Areas under ROC curves

| GaP | Panama | QL-CTR |
|-----|--------|--------|
| 0.82 | 0.72 | 0.80 |

Table 2: CTR lift of GaP over Panama

| View recall | 1% | 1-5% avg. | 5% | 5-20% avg. |
|-------------|-----|-----------|-----|------------|
| CTR lift | 0.96 | 0.86 | 0.93 | 0.68 |

## 4 Behavioral targeting

For the BT application, we adopt the first approach to CTR prediction as described in Section 2.1. The number of clicks and views for a given ad are predicted separately and a CTR estimator is constructed as in Eq. (8). Moreover, the granular nature of GaP allows for significant flexibility in the way prediction can be done, as we describe next.

### 4.1 Prediction with different granularity

We form the data matrix $F$ from historical user behavioral data at the granular level, including click and view counts for individual ads, as well as other explanatory variable features such as page views. This setup allows for per-ad CTR prediction, i.e., $p(\text{click}|\text{ad}, \text{user})$, given by Eq. (8). Per-category CTR prediction as does in previous BT systems, i.e., $p(\text{click}|\text{ad-category}, \text{user})$, can also be performed in this setup by marginalizing $\Lambda$ over categories:

$$\widehat{\text{CTR}}_{cj} = \left( \left( \sum_{i \in c} \Lambda_{\text{click}(i)} \right) \mathbf{x}_j + \delta \right) / \left( \left( \sum_{i \in c} \Lambda_{\text{view}(i)} \right) \mathbf{x}_j + \eta \right), \tag{13}$$

where $c$ denotes a category and $i \in c$ is defined by ad categorization.

The modeling was implemented in a distributed fashion using Hadoop. As discussed in Section 3.3, the EM algorithm can be parallelized efficiently by exploiting user data locality, particularly in the MapReduce [15] framework. However, compared with the scaling approach adopted by the SS implementation, the large-scale parallelization paradigm typically cannot support complex operations as efficient, such as performing sparse matrix multiplication by three-level nested loops in Java.

## 4.2 Experiments

The data matrix $F$ was formed to contain rows for all ad clicks and views, as well as page views with frequency above a threshold of 100. The counts were aggregated over a two-week period of time and from 32 buckets of users. This setup resulted in 170K features comprised of 120K ad clicks or views, and 50K page views, which allows the model matrix $\Lambda$ to fit well in memory. The number of users was about 40M. We set the latent inner dimension $d = 20$. We ran 13 EM iterations where each iteration alternated 3 E-steps with one M-step. Prediction accuracy was evaluated using data from the next day following the training period, and measured by the area under the ROC curve.

We first compared per-ad prediction (Eq. (8)) with per-category prediction (Eq. (13)), and obtained the ROC areas of 95% and 70%, respectively. One latest technology used Poisson regression for per-category modeling and yielded an average ROC area of 83% [11]. This shows that capturing intra-category structure by factor modeling can result in substantial improvement over the state-of-the-art of BT. We also measured the effect of the latent dimension on the model performance by varying $d = 10$ to $100$, and observed that per-ad prediction is insensitive to the latent dimension with all ROC areas in the range of $[95\%, 96\%]$, whereas per-category prediction benefits from larger inner dimensions. Finally, to verify the scalability of our parallel implementation, we increased the size of training data from 32 to 512 user buckets. The experiments were run on a 250-node Hadoop cluster. As shown in Table 3, the running time scales sub-linearly with the number of users.

Table 3: Run-time vs. number of user buckets

| Number of buckets | 32 | 64 | 128 | 512 |
|---|---|---|---|---|
| Run-time (hours) | 11.2 | 18.6 | 31.7 | 79.8 |

Surprisingly though, the running time for 32 buckets with a 250-node cluster is no less than a single-node yet highly efficient implementation as analyzed in Section 3.3 (after accounting for the different factors of users $4\times$, latent dimension $2\times$, and EM iterations $13/15$), with a similar 100 Mflops. Actually, the same pattern has been found in one previous large-scale learning task [11]. We argue that large-scale parallelization is not necessarily the best way, nor the only way, to deal with scaling; but in fact implementation issues (such as cache efficiency, number of references, data encapsulation) still cause orders-of-magnitude differences in performance and can more than overwhelm the additional nodes. The right principle of scaling up should start with single node and achieve above 100 Mflops with sparse arithmetic operations.

## 5 Discussion

GaP is a dimensionality reduction algorithm. The low-dimensional latent space allows scalable and efficient learning and prediction, and hence making the algorithm practically appealing for web-scale data like in SS and BT. GaP is also a smoothing algorithm, which yields smoothed click prediction. This addresses the data sparseness issue that is typically present in click-through data. Moreover, GaP builds personalization into ad targeting, by profiling a user as a vector of latent variables. The latent dimensions are inferred purely from data, with the objective to maximize the data likelihood or the capability to predict target events. Furthermore, position of ad impression has a significant impact on CTR. GaP factorization with one inner dimension gives a statistically sound approach to estimating the positional prior. Finally, the GaP-derived latent low-dimensional representation of user can be used as a valuable input to other applications and products, such as user clustering, collaborative filtering, content match, and algorithmic search.

# References

[1] http://hadoop.apache.org/.

[2] http://www.mathworks.com/products/matlab/.

[3] http://www.netlib.org/lapack/.

[4] D. M. Blei, A. Y. Ng, and M. I. Jordan. Latent Dirichlet allocation. *The Journal of Machine Learning Research*, 3:993–1022, 2003.

[5] A. Broder, M. Fontoura, V. Josifovski, and L. Riedel. A semantic approach to contextual advertising. *ACM Conference on Information Retrieval (SIGIR 2007)*, pages 559–566, 2007.

[6] J. F. Canny. GaP: a factor model for discrete data. *ACM Conference on Information Retrieval (SIGIR 2004)*, pages 122–129, 2004.

[7] J. F. Canny, S. Zhong, S. Gaffney, C. Brower, P. Berkhin, and G. H. John. Granular data for behavioral targeting. *U.S. Patent Application 20090006363*.

[8] D. Chakrabarti, D. Agarwal, and V. Josifovski. Contextual advertising by combining relevance with click feedback. *International World Wide Web Conference (WWW 2008)*, pages 417–426, 2008.

[9] O. Chapelle and Y. Zhang. A dynamic Bayesian network click model for web search ranking. *International World Wide Web Conference (WWW 2009)*, pages 1–10, 2009.

[10] Y. Chen, D. Pavlov, P. Berkhin, and J. F. Canny. Large-scale behavioral targeting for advertising over a network. *U.S. Patent Application 12/351,749*, filed: Jan 09, 2009.

[11] Y. Chen, D. Pavlov, and J. F. Canny. Large-scale behavioral targeting. *ACM Conference on Knowledge Discovery and Data Mining (KDD 2009)*, 2009.

[12] C. Y. Chung, J. M. Koran, L.-J. Lin, and H. Yin. Model for generating user profiles in a behavioral targeting system. *U.S. Patent 11/394,374*, filed: Mar 29, 2006.

[13] M. Ciaramita, V. Murdock, and V. Plachouras. Online learning from click data for sponsored search. *International World Wide Web Conference (WWW 2008)*, pages 227–236, 2008.

[14] N. Craswell, O. Zoeter, M. Taylor, and B. Ramsey. An experimental comparison of click position-bias models. *Web Search and Web Data Mining (WSDM 2008)*, pages 87–94, 2008.

[15] J. Dean and S. Ghemawat. Mapreduce: Simplified data processing on large clusters. *Communications of the ACM*, 51(1):107–113, 2008.

[16] S. Deerwester, S. T. Dumais, G. W. Furnas, T. K. Landauer, and R. Harshman. Indexing by latent semantic analysis. *Journal of the American Society for Information Science*, 41(6):391–407, 1990.

[17] D. C. Fain and J. O. Pedersen. Sponsored search: a brief history. *Bulletin of the American Society for Information Science and Technology*, 32(2):12–13, 2006.

[18] T. Hofmann. Unsupervised learning by probabilistic latent semantic analysis. *Machine Learning*, 42(1-2):177–196, 2001.

[19] A. Hyvärinen. Fast and robust fixed-point algorithms for independent component analysis. *IEEE Transactions on Neural Networks*, 10(3):626–634, 1999.

[20] I. T. Jolliffe. *Principal Component Analysis*. Springer, 2002.

[21] A. Lacerda, M. Cristo, M. A. Gonçalves, W. Fan, N. Ziviani, and B. Ribeiro-Neto. Learning to advertise. *ACM Conference on Information Retrieval (SIGIR 2006)*, pages 549–556, 2006.

[22] D. D. Lee and H. S. Seung. Algorithms for non-negative matrix factorization. *Advances in Neural Information Processing Systems (NIPS 2000)*, 13:556–562, 2000.

[23] S. Pandey and C. Olston. Handling advertisements of unknown quality in search advertising. *Advances in Neural Information Processing Systems (NIPS 2006)*, 19:1065–1072, 2006.

[24] F. Radlinski and T. Joachims. Minimally invasive randomization for collecting unbiased preferences from clickthrough logs. *National Conference on Artificial Intelligence (AAAI 2006)*, pages 1406–1412, 2006.

[25] M. Richardson, E. Dominowska, and R. Ragno. Predicting clicks: estimating the click-through rate for new ads. *International World Wide Web Conference (WWW 2007)*, pages 521–530, 2007.

[26] D. A. Spielman and S.-H. Teng. Smoothed analysis of algorithms: Why the simplex algorithm usually takes polynomial time. *Journal of the ACM*, 51(3):385–463, 2004.

